# Generalized Lasso based Approximation of Sparse Coding for Visual Recognition

**Nobuyuki Morioka**
The University of New South Wales & NICTA
Sydney, Australia
nmorioka@cse.unsw.edu.au

**Shin'ichi Satoh**
National Institute of Informatics
Tokyo, Japan
satoh@nii.ac.jp

## Abstract

Sparse coding, a method of explaining sensory data with as few dictionary bases as possible, has attracted much attention in computer vision. For visual object category recognition, $\ell_1$ regularized sparse coding is combined with the spatial pyramid representation to obtain state-of-the-art performance. However, because of its iterative optimization, applying sparse coding onto every local feature descriptor extracted from an image database can become a major bottleneck. To overcome this computational challenge, this paper presents "Generalized Lasso based Approximation of Sparse coding" (GLAS). By representing the distribution of sparse coefficients with slice transform, we fit a piece-wise linear mapping function with the generalized lasso. We also propose an efficient post-refinement procedure to perform mutual inhibition between bases which is essential for an overcomplete setting. The experiments show that GLAS obtains a comparable performance to $\ell_1$ regularized sparse coding, yet achieves a significant speed up demonstrating its effectiveness for large-scale visual recognition problems.

## 1 Introduction

Recently, sparse coding [3, 18] has attracted much attention in computer vision research. Its applications range from image denoising [23] to image segmentation [17] and image classification [10, 24], achieving state-of-the-art results. Sparse coding interprets an input signal $\mathbf{x} \in \mathbb{R}^{D \times 1}$ with a sparse vector $\mathbf{u} \in \mathbb{R}^{K \times 1}$ whose linear combination with an overcomplete set of $K$ bases (i.e., $D \ll K$), also known as dictionary $\mathbf{B} \in \mathbb{R}^{D \times K}$, reconstructs the input as precisely as possible. To enforce sparseness on $\mathbf{u}$, the $\ell_1$ norm is a popular choice due to its computational convenience and its interesting connection with the NP-hard $\ell_0$ norm in compressed sensing [2]. Several efficient $\ell_1$ regularized sparse coding algorithms have been proposed [4, 14] and are adopted in visual recognition [10, 24]. In particular, Yang et al. [24] compute the spare codes of many local feature descriptors with sparse coding. However, due to the $\ell_1$ norm being non-smooth convex, the sparse coding algorithm needs to optimize iteratively until convergence. Therefore, the local feature descriptor coding step becomes a major bottleneck for large-scale problems like visual recognition.

The goal of this paper is to achieve state-of-the-art performance on large-scale visual recognition that is comparable to the work of Yang et al. [24], but with a significant improvement in efficiency. To this end, we propose "Generalized Lasso based Approximation of Sparse coding", GLAS for short. Specifically, we encode the distribution of each dimension in sparse codes with the slice transform representation [9] and learn a piece-wise linear mapping function with the generalized lasso to obtain the best fit [21] to approximate $\ell_1$ regularized sparse coding. We further propose an efficient post-refinement procedure to capture the dependency between overcomplete bases. The effectiveness of our approach is demonstrated with several challenging object and scene category datasets, showing a comparable performance to Yang et al. [24] and performing better than other fast algorithms that obtain sparse codes [22]. While there have been several supervised dictionary

learning methods for sparse coding to obtain more discriminative sparse representations [16, 25], they have not been evaluated on visual recognition with many object categories due to its computational challenges. Furthermore, Ranzato et al. [19] have empirically shown that unsupervised learning of visual features can obtain a more general and effective representation. Therefore, in this paper, we focus on learning a fast approximation of sparse coding in an unsupervised manner.

The paper is organized as follows: Section 2 reviews some related work including the linear spatial pyramid combined with sparse coding and other fast algorithms to obtain sparse codes. Section 3 presents GLAS. This is followed by the experimental results on several challenging categorization datasets in Section 4. Section 5 concludes the paper with discussion and future work.

## 2 Related Work

### 2.1 Linear Spatial Pyramid Matching Using Sparse Coding

This section reviews the linear spatial pyramid matching based on sparse coding by Yang et al. [24]. Given a collection of $N$ local feature descriptors randomly sampled from training images $\mathbf{X} = [\mathbf{x}_1, \mathbf{x}_2, \ldots, \mathbf{x}_N] \in \mathbb{R}^{D \times N}$, an over-complete dictionary $\mathbf{B} = [\mathbf{b}_1, \mathbf{b}_2, \ldots, \mathbf{b}_K] \in \mathbb{R}^{D \times K}$ is learned by

$$\min_{\mathbf{B}, \mathbf{U}} \sum_{i=1}^{N} \|\mathbf{x}_i - \mathbf{B}\mathbf{u}_i\|_2^2 + \lambda \|\mathbf{u}_i\|_1 \quad s.t. \quad \|\mathbf{b}_k\|_2^2 \leq 1, k = 1, 2, \ldots, K. \tag{1}$$

The cost function above is a combination of the reconstruction error and the $\ell_1$ sparsity penalty which is controlled by $\lambda$. The $\ell_2$ norm on each $\mathbf{b}_k$ is constrained to be less than or equal to 1 to avoid a trival solution. Since both $\mathbf{B}$ and $[\mathbf{u}_1, \mathbf{u}_2, \ldots, \mathbf{u}_N]$ are unknown a priori, an alternating optimization technique is often used [14] to optimize over the two parameter sets.

Under the spatial pyramid matching framework, each image is divided into a set of sub-regions $\mathbf{r} = [r_1, r_2, \ldots, r_R]$. For example, if $1 \times 1$, $2 \times 2$ and $4 \times 4$ partitions are used on an image, we have 21 sub-regions. Then, we compute the sparse solutions of all local feature descriptors, denoted as $\mathbf{U}_{r_j}$, appearing in each sub-region $r_j$ by

$$\min_{\mathbf{U}_{r_j}} \|\mathbf{X}_{r_j} - \mathbf{B}\mathbf{U}_{r_j}\|_2^2 + \lambda \|\mathbf{U}_{r_j}\|_1. \tag{2}$$

The sparse solutions are max pooled for each sub-region and concatenated with other sub-regions to build a statistic of the image by

$$\mathbf{h} = [\max(|\mathbf{U}_{r_1}|)^\top, \max(|\mathbf{U}_{r_2}|)^\top, \ldots, \max(|\mathbf{U}_{r_R}|)^\top]^\top, \tag{3}$$

where max(.) is a function that finds the maximum value at each row of a matrix and returns a column vector. Finally, a linear SVM is trained on a set of image statistics for classification.

The main advantage of using sparse coding is that state-of-the-art results can be achieved with a simple linear classifier as reported in [24]. Compared to kernel-based methods, this dramatically speeds up training and testing time of the classifier. However, the step of finding a sparse code for each local descriptor with sparse coding now becomes a major bottleneck. Using the efficient sparse coding algorithm based on feature-sign search [14], the time to compute the solution for one local descriptor $\mathbf{u}$ is $O(KZ)$ where $Z$ is the number of non-zeros in $\mathbf{u}$. This paper proposes an approximation method whose time complexity reduces to $O(K)$. With the post-refinement procedure, its time complexity is $O(K + Z^2)$ which is still much lower than $O(KZ)$.

### 2.2 Predictive Sparse Decomposition

Predictive sparse decomposition (PSD) described in [10, 11] is a feedforward network that applies a non-linear mapping function on linearly transformed input data to match the optimal sparse coding solution as accurate as possible. Such feedfoward network is defined as: $\hat{\mathbf{u}}_i = \mathbf{G}g(\mathbf{W}\mathbf{x}_i, \theta)$, where $g(z, \theta)$ denotes a non-linear parametric mapping function which can be of any form, but to name a few there are hyperbolic tangent, $\tanh(z + \theta)$ and soft shrinkage, $\mathrm{sign}(z)\max(|z| - \theta, 0)$. The function is applied to linearly transformed data $\mathbf{W}\mathbf{x}_i$ and subsequently scaled by a diagonal matrix

**G**. Given training samples $\{\mathbf{x}_i\}_{i=1}^N$, the parameters can be estimated either jointly or separately from the dictionary **B**. When learning jointly, we minimize the cost function given below:

$$\min_{\mathbf{B},\mathbf{G},\mathbf{W},\theta,\mathbf{U}} \sum_{i=1}^N \|\mathbf{x}_i - \mathbf{B}\mathbf{u}_i\|_2^2 + \lambda\|\mathbf{u}_i\|_1 + \gamma\|\mathbf{u}_i - \mathbf{G}g(\mathbf{W}\mathbf{x}_i,\theta)\|_2^2. \tag{4}$$

When learning separately, **B** and **U** are obtained with Eqn. (1) first. Then, other remaining parameters **G**, **W** and $\theta$ are estimated by solving the last term of Eqn. (4) only. Gregor and LeCun [7] have later proposed a better, but iterative approximation scheme for $\ell_1$ regularized sparse coding.

One downside of the parametric approach is its accuracy is largely dependent on how well its parametric function fits the target statistical distribution, as argued by Hel-Or and Shaked [9]. This paper explores a non-parametric approach which can fit any distribution as long as data samples available are representative. The advantage of our approach over the parametric approach is that we do not need to seek an appropriate parametric function for each distribution. This is particularly useful in visual recognition that uses multiple feature types, as it automatically estimates the function form for each feature type from data. We demonstrate this with two different local descriptor types in our experiments.

## 2.3 Locality-constrained Linear Coding

Another notable work that overcomes the bottleneck of the local descriptor coding step is locality-constrained linear coding (LLC) proposed by Wang et al. [22], a fast version of local coordinate coding [26]. Given a local feature descriptor $\mathbf{x}_i$, LLC searches for $M$ nearest dictionary bases of each local descriptor $\mathbf{x}_i$ and these nearest bases stacked in columns are denoted as $\mathbf{B}_{\phi_i} \in \mathbb{R}^{D \times M}$ where $\phi_i$ indicates the index list of the bases. Then, the coefficients $\mathbf{u}_{\phi_i} \in \mathbb{R}^{M \times 1}$ whose linear combination with $\mathbf{B}_{\phi_i}$ reconstructs $\mathbf{x}_i$ is solved by:

$$\min_{\mathbf{u}_{\phi_i}} \|\mathbf{x}_i - \mathbf{B}_{\phi_i}\mathbf{u}_{\phi_i}\|_2^2 \quad s.t. \quad \mathbf{1}^\top \mathbf{u}_{\phi_i} = 1. \tag{5}$$

This is the least squares problem which can be solved quite efficiently. The final sparse code $\mathbf{u}_i$ is obtained by setting its elements indexed at $\phi_i$ to $\mathbf{u}_{\phi_i}$. The time complexity of LLC is $O(K + M^2)$. This excludes the time required to find $M$ nearest neighbours. While it is fast, the resulting sparse solutions obtained are not as discriminative as the ones obtained by sparse coding. This may be due to the fact that $M$ is fixed across all local feature descriptors. Some descriptors may need more bases for accurate representation and others may need less bases for more distinctiveness. In contrast, the number of bases selected with our post-refinement procedure to handle the mutual inhibition is different for each local descriptor.

## 3 Generalized Lasso based Approximation of Sparse Coding

This section describes GLAS. We first learn a dictionary from a collection of local feature descriptors as given Eqn. (1). Then, based on slice transform representation, we fit a piece-wise linear mapping function with the generalized lasso to approximate the optimal sparse solutions of the local feature descriptors under $\ell_1$ regularized sparse coding. Finally, we propose an efficient post-refinement procedure to perform the mutual inhibition.

### 3.1 Slice Transform Representation

Slice transform representation is introduced as a way to discretize a function space so to fit a piece-wise linear function for the purpose of image denoising by Hel-Or and Shaked [9]. This is later adopted by Adler et al. [1] for single image super resolution. In this paper, we utilise the representation to approximate sparse coding to obtain sparse codes for local feature descriptor as fast as possible.

Given a local descriptor $\mathbf{x}$, we can linearly combine with $\mathbf{B}^\top$ to obtain $\mathbf{z}$. For the moment, we just consider one dimension of $\mathbf{z}$ denoted as $z$ which is a real value and lies in a half open interval of $[a, b)$. The interval is divided into $Q - 1$ equal-sized bins whose boundaries form a vector $\mathbf{q} = [q_1, q_2, \ldots, q_Q]^\top$ such that $a = q_1 < q_2 \cdots < q_Q = b$.

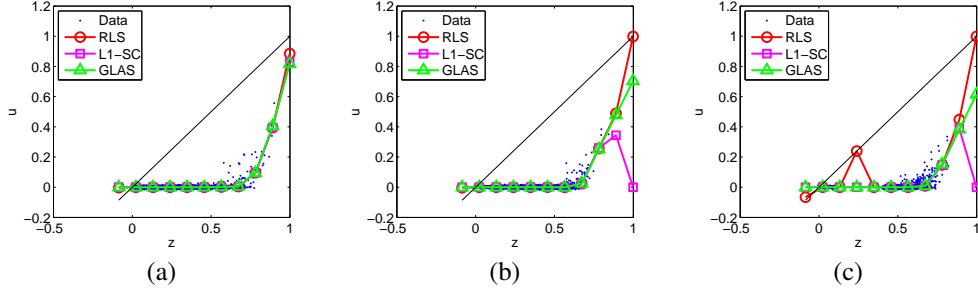

Figure 1: Different approaches to fit a piece-wise linear mapping function. Regularized least squares (RLS) in red (see Eqn. (8)). $\ell_1$-regularized sparse coding (L1-SC) in magenta (see Eqn. (9)). GLAS in green (see Eqn. (10)). (a) All three methods achieving a good fit. (b) A case when L1-SC fails to extrapolate well at the end and RLS tends to align itself to $\mathbf{q}$ in black. (c) A case when data samples at around 0.25 are removed artificially to illustrate that RLS fails to interpolate as no neighoring prior is used. In contrast, GLAS can both interpolate and extrapolate well in the case of missing or noisy data.

The interval into which the value of $z$ falls is expressed as: $\pi(z) = j$ if $z \in [q_{j-1}, q_j)$, and its corresponding residue is calculated by: $r(z) = \frac{z - q_{\pi(z)-1}}{q_{\pi(z)} - q_{\pi(z)-1}}$.

Based on the above, we can re-express $z$ as

$$z = (1 - r(z))q_{\pi(z)-1} + r(z)q_{\pi(z)} = S_{\mathbf{q}}(z)\mathbf{q}, \tag{6}$$

where $S_{\mathbf{q}}(z) = [0, \ldots, 0, 1 - r(z), r(z), 0, \ldots, 0]$.

If we now come back to the multivariate case of $\mathbf{z} = \mathbf{B}^\top \mathbf{x}$, then we have the following: $\mathbf{z} = [S_{\mathbf{q}}(z_1)\mathbf{q}, S_{\mathbf{q}}(z_2)\mathbf{q}, \ldots, S_{\mathbf{q}}(z_K)\mathbf{q}]^\top$, where $z_k$ implies the $k^{th}$ dimension of $\mathbf{z}$. Then, we replace the boundary vector $\mathbf{q}$ with $\mathbf{p} = \{\mathbf{p}_1, \mathbf{p}_2, \ldots, \mathbf{p}_K\}$ such that resulting vector approximates the optimal sparse solution of $\mathbf{x}$ obtained by $\ell_1$ regularized sparse coding as much as possible. This is written as

$$\hat{\mathbf{u}} = [S_{\mathbf{q}}(z_1)\mathbf{p}_1, S_{\mathbf{q}}(z_2)\mathbf{p}_2, \ldots, S_{\mathbf{q}}(z_K)\mathbf{p}_K]^\top. \tag{7}$$

Hel-Or and Shaked [9] have formulated the problem of learning each $\mathbf{p}_k$ as regularized least squares either independently in a transform domain or jointly in a spatial domain. Unlike their setting, we have significantly large number of bases which makes joint optimization of all $\mathbf{p}_k$ difficult. Moreover, since we are interested in approximating the sparse solutions which are in the transform domain, we learn each $\mathbf{p}_k$ independently. Given $N$ local descriptors $\mathbf{X} = [\mathbf{x}_1, \mathbf{x}_2, \ldots, \mathbf{x}_N] \in \mathbb{R}^{D \times N}$ and their corresponding sparse solutions $\mathbf{U} = [\mathbf{u}_1, \mathbf{u}_2, \ldots, \mathbf{u}_N] = [\mathbf{y}_1, \mathbf{y}_2, \ldots, \mathbf{y}_K]^\top \in \mathbb{R}^{K \times N}$ obtained with $\ell_1$ regularized sparse coding, we have an optimization problem given as

$$\min_{\mathbf{p}_k} \|\mathbf{y}_k - \mathbf{S}_k \mathbf{p}_k\|_2^2 + \alpha \|\mathbf{q} - \mathbf{p}_k\|_2^2, \tag{8}$$

where $\mathbf{S}_k = S_{\mathbf{q}}(\mathbf{z}_k)$. The regularization of the second term is essential to avoid singularity when computing the inverse and its consequence is that $\mathbf{p}_k$ is encouraged to align itself to $\mathbf{q}$ when not many data samples are available. This might have been a reasonable prior for image denoising [9], but not desirable for the purpose of approximating sparse coing, as we would like to suppress most of the coefficients in $\mathbf{u}$ to zero. Figure 1 shows the distribution of one dimension of sparse coefficients $z$ obtained from a collection of SIFT descriptors and $\mathbf{q}$ does not look similar to the distribution. This motivates us to look at the generalized lasso [21] as an alternative for obtaining a better fit of the distribution of the coefficients.

## 3.2 Generalized Lasso

In the previous section, we have argued that regularized least squares stated in Eqn. (8) does not give the desired result. Instead most intervals need to be set to zero. This naturally leads us to consider $\ell_1$ regularized sparse coding also known as the lasso which is formulated as:

$$\min_{\mathbf{p}_k} \|\mathbf{y}_k - \mathbf{S}_k \mathbf{p}_k\|_2^2 + \alpha \|\mathbf{p}_k\|_1. \tag{9}$$

However, the drawback of this is that the learnt piece-wise linear function may become unstable in cases when training data is noisy or missing as illustrated in Figure 1 (b) and (c). It turns out $\ell_1$ trend filtering [12], generally known as the generalized lasso [21], can overcome this problem. This is expressed as

$$\min_{\mathbf{p}_k} \|\mathbf{y}_k - \mathbf{S}_k \mathbf{p}_k\|_2^2 + \alpha\|\mathbf{D}\mathbf{p}_k\|_1, \tag{10}$$

where $\mathbf{D} \in \mathbb{R}^{(Q-2)\times Q}$ is referred to as a penalty matrix and defined as

$$\mathbf{D} = \begin{bmatrix} -1 & 2 & -1 & & \\ & -1 & 2 & -1 & \\ & & \ddots & \ddots & \ddots \\ & & & -1 & 2 & -1 \end{bmatrix}. \tag{11}$$

To solve the above optimization problem, we can turn it into the sparse coding problem [21]. Since $\mathbf{D}$ is not invertible, the key is to augment $\mathbf{D}$ with $\mathbf{A} \in \mathbb{R}^{2\times Q}$ to build a square matrix $\tilde{\mathbf{D}} = [\mathbf{D}; \mathbf{A}] \in \mathbb{R}^{Q\times Q}$ such that $\text{rank}(\tilde{\mathbf{D}}) = Q$ and the rows of $\mathbf{A}$ are orthogonal to the rows of $\mathbf{D}$. To satisfy such constraints, $\mathbf{A}$ can for example be set to $[1, 2, \ldots, Q; 2, 3, \ldots, Q + 1]$. If we let $\theta = [\theta_1; \theta_2] = \tilde{\mathbf{D}}\mathbf{p}_k$ where $\theta_1 = \mathbf{D}\mathbf{p}_k$ and $\theta_2 = \mathbf{A}\mathbf{p}_k$, then $\mathbf{S}_k\mathbf{p}_k = \mathbf{S}_k\tilde{\mathbf{D}}^{-1}\theta = \mathbf{S}_{k1}\theta_1 + \mathbf{S}_{k2}\theta_2$. After some substitutions, we see that $\theta_2$ can be solved by: $\theta_2 = (\mathbf{S}_{k2}^\top\mathbf{S}_{k2})^{-1}\mathbf{S}_{k2}^\top(\mathbf{y}_k - \mathbf{S}_{k1}\theta_1)$, given $\theta_1$ is solved already. Now, to solve $\theta_1$, we have the following sparse coding problem:

$$\min_{\theta_1} \|(I - \mathbf{P})\mathbf{y}_k - (I - \mathbf{P})\mathbf{S}_{k1}\theta_1\|_2^2 + \alpha\|\theta_1\|_1, \tag{12}$$

where $\mathbf{P} = \mathbf{S}_{k2}(\mathbf{S}_{k2}^\top\mathbf{S}_{k2})^{-1}\mathbf{S}_{k2}^\top$. Having computed both $\theta_1$ and $\theta_2$, we can recover the solution of $\mathbf{p}_k$ by $\tilde{\mathbf{D}}^{-1}\theta$. Further details can be found in [21].

Given the learnt $\mathbf{p}$, we can approximate sparse solution of $\mathbf{x}$ by Eqn. (7). However, explicitly computing $S_{\mathbf{q}}(\mathbf{z})$ and multiplying it by $\mathbf{p}$ is somewhat redundant. Thus, we can alternatively compute each component of $\hat{\mathbf{u}}$ as follows:

$$\hat{u}_k = (1 - r(z_k)) \times \mathbf{p}_k(\pi(z_k) - 1) + r(z_k) \times \mathbf{p}_k(\pi(z_k)), \tag{13}$$

whose time complexity becomes $O(K)$. In Eqn. (13), since we are essentially using $\mathbf{p}_k$ as a lookup table, the complexity is independent from $Q$. This is followed by $\ell_1$ normalization on $\hat{\mathbf{u}}$.

While $\hat{\mathbf{u}}$ can readily be used for the spatial max pooling as stated in Eqn. (3), it does not yet capture any "explaining away" effect where the corresponding coefficients of correlated bases are mutually inhibited to remove redundancy. This is because each $\mathbf{p}_k$ is estimated independently in the transform domain [9]. In the next section, we propose an efficient post-refinement technique to mutually inhibit between the bases.

### 3.3 Capturing Dependency Between Bases

To handle the mutual inhibition between overcomplete bases, this section explains how to refine the sparse codes by solving regularized least squares on a significantly small active basis set. Given a local descriptor $\mathbf{x}$ and its initial sparse code $\hat{\mathbf{u}}$ estimated with above method, we set the non-zero components of the code to be active. By denoting a set of these active components as $\phi$, we have $\hat{\mathbf{u}}_\phi$ and $\mathbf{B}_\phi$ which are the subsets of the sparse code and dictionary bases respectively. The goal is to compute the refined code of $\hat{\mathbf{u}}_\phi$ denoted as $\hat{\mathbf{v}}_\phi$ such that $\mathbf{B}_\phi\mathbf{v}_\phi$ reconstructs $\mathbf{x}_i$ as accurately as possible. We formulate this as regularised least squares given below:

$$\min_{\hat{\mathbf{v}}_\phi} \|\mathbf{x} - \mathbf{B}_\phi\hat{\mathbf{v}}_\phi\|_2^2 + \beta\|\hat{\mathbf{v}}_\phi - \hat{\mathbf{u}}_\phi\|_2^2, \tag{14}$$

where $\beta$ is the weight parameter of the regularisation. This is convex and has the following analytical solution: $\hat{\mathbf{v}}_\phi = (\mathbf{B}_\phi^\top\mathbf{B}_\phi + \beta I)^{-1}(\mathbf{B}_\phi^\top\mathbf{x} + \beta\hat{\mathbf{u}}_\phi)$.

The intuition behind the above formulation is that the initial sparse code $\hat{\mathbf{u}}$ is considered as a good starting point for refinement to further reduce the reconstruction error by allowing redundant bases to compete against each other. Empirically, the number of active components for each $\hat{\mathbf{u}}$ is substantially small compared to the whole basis set. Hence, a linear system to be solved becomes much smaller

**SIFT (128 Dim.)** [15]

| Methods | KM | LLC [22] | PSD [11] | SC [24] | **GLAS** | **GLAS+** |
|---------|-----|----------|----------|---------|----------|-----------|
| 15 Train | 55.5±1.2 | 62.7±1.0 | 64.0±1.2 | 65.2±1.2 | 64.4±1.2 | 65.1±1.1 |
| 30 Train | 63.0±1.2 | 69.6±0.8 | 70.6±0.9 | 71.6±0.7 | 71.6±1.0 | 72.3±0.7 |
| Time (sec) | 0.06 | 0.25 | 0.06 | 3.53 | 0.15 | 0.23 |

**Local Self-Similarity (30 Dim.)** [20]

| Methods | KM | LLC [22] | PSD [11] | SC [24] | **GLAS** | **GLAS+** |
|---------|-----|----------|----------|---------|----------|-----------|
| 15 Train | 60.1±1.3 | 62.4±0.8 | 59.7±0.8 | 64.8±0.9 | 62.3±1.2 | 63.8±0.9 |
| 30 Train | 63.0±1.2 | 69.7±1.3 | 67.2±0.9 | 72.5±1.6 | 69.8±1.4 | 71.0±1.1 |
| Time (sec) | 0.05 | 0.24 | 0.05 | 1.97 | 0.13 | 0.18 |

Table 1: Recognition accuracy on Caltech-101. The dictionary sizes for all methods are set to 1024. We also report the time taken to process 1000 local descriptors for each method.

which is computationally cheap. We also make sure that we do not deviate too much from the initial solution by introducing the regularization on $\hat{\mathbf{v}}_\phi$. This refinement procedure may be similar to LLC [22]. However, in our case, we do not preset the number of active bases and determine by non-zero components of $\hat{\mathbf{u}}$. More importantly, we base our final solution on $\hat{\mathbf{u}}$ and do not perform nearest neighbor search. With this refinement procedure, the total time complexity becomes $O(K + Z^2)$. We refer GLAS with this post-refinement procedure as GLAS+.

## 4 Experimental Results

This section evaluates GLAS and GLAS+ on several challenging categorization datasets. To learn the mapping function, we have used 50,000 local descriptors as data samples. The parameters $Q$, $\alpha$ and $\beta$ are fixed to 10, 0.1 and 0.25 respectively for all experiments, unless otherwise stated. For comparison, we have implemented methods discussed in Section 2. SC is our re-implementation of Yang et al. [24]. LLC is locality-constrained linear coding proposed by Wang et al. [22]. The number of nearest neighors to consider is set to 5. PSD is predictive sparse decomposition [11]. Shrinkage function is used as its parametric mapping function. We also include KM which builds its codebook with $k$-means clustering and adopts hard-assignment as its local descriptor coding.

For all methods, exactly the same local feature descriptors, spatial max pooling technique and linear SVM are used to only compare the difference between the local feature descriptor coding techniques. As for the descriptors, SIFT [15] and Local Self-Similarity [20] are used. SIFT is a histogram of gradient directions computed over an image patch - capturing appearance information. We have sampled a 16×16 patch at every 8 pixel step. In contrast, Local Self-Similarity computes correlation between a small image patch of interest and its surrounding region which captures the geometric layout of a local region. Spatial max pooling with $1 \times 1$, $2 \times 2$ and $4 \times 4$ image partitions is used. The implementation is all done in MATLAB for fair comparison.

### 4.1 Caltech-101

The Caltech-101 dataset [5] consists of 9144 images which are divided into 101 object categories. The images are scaled down to $300 \times 300$ preserving their aspect ratios. We train with 15/30 images per class and test with 15 images per class. The dictionary size of each method is set to 1024 for both SIFT and Local Self-Similarity.

The results are averaged over eight random training and testing splits and are reported in Table 1. For SIFT, GLAS+ is consistently better than GLAS demonstrating the effectiveness of mutual inhibition by the post-refinement procedure. Both GLAS and GLAS+ performs better than other fast algorithms that produces sparse codes. In addition GLAS and GLAS+ performs competitively against SC. In fact, GLAS+ is slightly better when 30 training images per class are used. While sparse codes for both GLAS and GLAS+ are learned from the solutions of SC, the approximated codes are not exactly the same as the ones of SC. Moreover, SC sometimes produces unstable codes due to the non-smooth convex property of $\ell_1$ norm as previously observed in [6]. In contrast, GLAS+

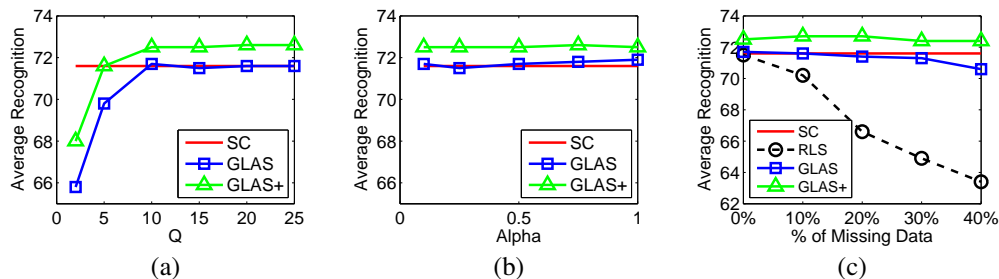

Figure 2: (a) $Q$, the number of bins to quantize the interval of each sparse code component. (b) $\alpha$, the parameter that controls the weight of the norm used for the generalized lasso. (c) When some data samples are missing GLAS is more robust than regularized least squares given in Eqn. (8).

approximates its sparse codes with a relatively smooth piece-wise linear mapping function learned with the generalized lasso (note that the $\ell_1$ norm penalizes on changes in the shape of the function) and performs smooth post-refinement. We suspect these differences may be contributing to the slightly better results of GLAS+ on this dataset.

Although PSD performs quite close to GLAS for SIFT, this is not the case for Local Self-Similarity. GLAS outperforms PSD probably due to the distribution of sparse codes is not captured well by a simple shrinkage function. Therefore, GLAS might be more effective for a wide range of distributions. This is useful for recognition using multiple feature types where speed is critical. GLAS performs worse than SC, but GLAS+ closes the gap between GLAS and SC. We suspect that due to Local Self-Similarity (30 dim.) being relatively low-dimensional than SIFT (128 dim.), the mutual inhibition becomes more important. This might also explain why LLC has performed reasonably well for this descriptor.

Table 1 also reports computational time taken to process 1000 local descriptors for each method. GLAS and GLAS+ are slower than KM and PSD, but are slightly faster than LLC and significantly faster than SC. This demonstrates the practical importance of our approach where competitive recognition results are achieved with fast computation.

Different values for $Q$, $\alpha$ and $\beta$ are evaluated one parameter at a time. Figure 2 (a) shows the results of different $Q$. The results are very stable after 10 bins. As sparse codes are computed by Eqn. (13), the time complexity is not affected by what $Q$ is chosen. Figure 2 (b) shows the results for different $\alpha$ which look very stable. We also observe similar stability for $\beta$.

We also validate if the generalized lasso given in Eqn. (10) is more robust than the regularized least squares solution given in Eqn. (8) when some data samples are missing. When learning each $\mathbf{q}_k$, we artificially remove data samples from an interval centered around a randomly sampled point, as also illustrated in Figure 1 (c). We evaluate with different numbers of data samples removed in terms of percentages of the whole data sample set. The results are shown in Figure 2 (c) where the performance of RLS significantly drops as the number of missing data is increased. However, both GLAS and GLAS+ are not affected that much.

### 4.2 Caltech-256

Caltech-256 [8] contains 30,607 images and 256 object categories in total. Like Caltech-101, we scale the images down to $300 \times 300$ preserving their aspect ratios. The results are averaged over eight random training and testing splits and are reported in Table 2. We use 25 testing images per class. This time, for SIFT, GLAS performs slightly worse than SC, but GLAS+ outperforms SC probably due to the same argument given in the previous experiments on Caltech-101. For Local Self-Similarity, results similar to Caltech-101 are obtained. The performance of PSD is close to KM and is outperformed by GLAS, suggesting the inadequate fitting of sparse codes. LLC performs slightly better than GLAS, but could not perform better than GLAS+. While SC performed the best, the performance of GLAS+ is quite close to SC. We also plot a graph of the computational time taken for each method with its achieved accuracy on SIFT and Local Self-Similarity in Figure 3 (a) and (b) respectively.

<div align="center">

**SIFT (128 Dim.)** [15]

| Methods | KM | LLC [22] | PSD [11] | SC [24] | **GLAS** | **GLAS+** |
|---------|----|---------|---------|--------|----------|-----------|
| 15 Train | 22.7±0.4 | 28.1±0.5 | 30.4±0.6 | 30.7±0.4 | 30.4±0.4 | 32.1±0.4 |
| 30 Train | 27.4±0.5 | 34.0±0.6 | 36.3±0.5 | 36.8±0.4 | 36.1±0.4 | 38.2±0.4 |

**Local Self-Similarity (30 Dim.)** [20]

| Methods | KM | LLC [22] | PSD [11] | SC [24] | **GLAS** | **GLAS+** |
|---------|----|---------|---------|--------|----------|-----------|
| 15 Train | 23.7±0.4 | 26.3±0.5 | 24.3±0.6 | 28.7±0.5 | 26.0±0.5 | 27.6±0.6 |
| 30 Train | 28.5±0.4 | 31.9±0.5 | 29.3±0.5 | 34.7±0.4 | 31.2±0.5 | 33.3±0.5 |

</div>

Table 2: Recognition accuracy on Caltech-256. The dictionary sizes are all set to 2048 for SIFT and 1024 for Local Self-Similarity.

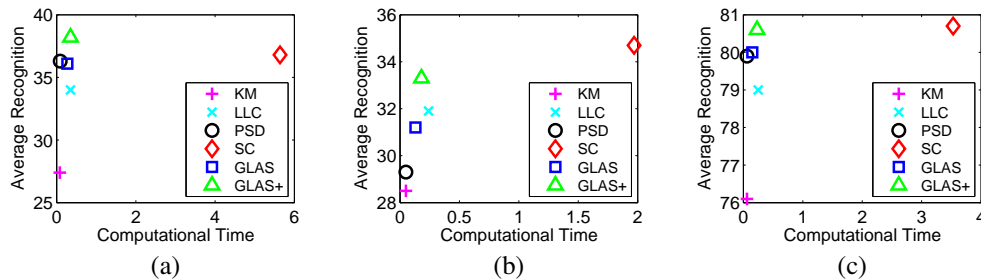

Figure 3: Plotting computational time vs. average recognition. (a) and (b) are SIFT and Local-Self Similarity respectively evaluated on Caltech-256 with 30 training images. The dictionary size is set to 2048. (c) is SIFT evaluated on 15 Scenes. The dictionary size is set to 1024.

### 4.3 15 Scenes

The 15 Scenes [13] dataset contains 4485 images divided into 15 scene classes ranging from indoor scenes to outdoor scenes. 100 training images per class are used for training and the rest for testing. We used SIFT to learn 1024 dictionary bases for each method. The results are plotted with computational time taken in Figure 3 (c). The result of GLAS+ (80.6%) are very similar to that of SC (80.7%), yet the former is significantly faster. In summary, we show that our approach works well on three different challenging datasets.

## 5   Conclusion

This paper has presented an approximation of $\ell_1$ sparse coding based on the generalized lasso called GLAS. This is further extended with the post-refinement procedure to handle mutual inhibition between bases which are essential in an overcomplete setting. The experiments have shown competitive performance of GLAS against SC and achieved significant computational speed up. We have also demonstrated that the effectiveness of GLAS on two local descriptor types, namely SIFT and Local Self-Similarity where LLC and PSD only perform well on one type. GLAS is not restricted to only approximate $\ell_1$ sparse coding, but should be applicable to other variations of sparse coding in general. For example, it may be interesting to try GLAS on Laplacian sparse coding [6] that achieves smoother sparse codes than $\ell_1$ sparse coding.

## Acknowledgment

NICTA is funded by the Australian Government as represented by the Department of Broadband, Communications and the Digital Economy and the Australian Research Council through the ICT Centre of Excellence program.

# References

[1] A. Adler, Y. Hel-Or, and M. Elad. A Shrinkage Learning Approach for Single Image Super-Resolution with Overcomplete Representations. In *ECCV*, 2010.

[2] D.L. Donoho. For Most Large Underdetermined Systems of Linear Equations the Minimal L1-norm Solution is also the Sparse Solution. *Communications on Pure and Applied Mathematics*, 2006.

[3] D.L. Donoho and M. Elad. Optimally sparse representation in general (nonorthogonal) dictionaries via L1 minimization. *PNAS*, 100(5):2197–2202, 2003.

[4] B. Efron, T. Hastie, I. Johnstone, and R. Tibshirani. Least Angle Regression. *Annals of Statistics*, 2004.

[5] L. Fei-Fei, R. Fergus, and P. Perona. Learning Generative Visual Models from Few Training Examples: An Incremental Bayesian Approach Tested on 101 Object Categories. In *CVPR Workshop*, 2004.

[6] S. Gao, W. Tsang, L. Chia, and P. Zhao. Local Features Are Not Lonely - Laplacian Sparse Coding for Image Classification. In *CVPR*, 2010.

[7] K. Gregor and Y. LeCun. Learning fast approximations of sparse coding. In *ICML*, 2010.

[8] G. Griffin, A. Holub, and P. Perona. Caltech-256 Object Category Dataset. *Technical Report, California Institute of Technology*, 2007.

[9] Y. Hel-Or and D. Shaked. A Discriminative Approach for Wavelet Denoising. *TIP*, 2008.

[10] K. Jarrett, K. Kavukcuoglu, M. Ranzato, and Y. LeCun. What is the Best Multi-Stage Architecture for Object Recognition. In *ICCV*, 2009.

[11] K Kavukcuoglu, M Ranzato, and Y Lecun. Fast inference in sparse coding algorithms with applications to object recognition. *Technical rRport CBLL-TR-2008-12-01, Computational and Biological Learning Lab, Courant Institute, NYU*, 2008.

[12] S.-J. Kim, K. Koh, S. Boyd, and D. Gorinevsky. L1 trend filtering. *SIAM Review*, 2009.

[13] S. Lazebnik, C. Schmid, and J. Ponce. Beyond Bags of Features: Spatial Pyramid Matching for Recognizing Natural Scene Categories. In *CVPR*, 2006.

[14] H. Lee, A. Battle, R. Raina, and A.Y. Ng. Efficient sparse coding algorithms. In *NIPS*, 2006.

[15] D.G. Lowe. Distinctive Image Features from Scale-Invariant Keypoints. *IJCV*, 2004.

[16] J. Mairal, F. Bach, J. Ponce, G. Sapiro, and A. Zisserman. Supervised Dictionary Learning. In *NIPS*, 2008.

[17] J. Mairal, M. Leordeanu, F. Bach, M. Hebert, and J. Ponce. Discriminative Sparse Image Models for Class-Specific Edge Detection and Image Interpretation. In *ECCV*, 2008.

[18] B.A. Olshausen and D.J. Field. Sparse coding with an overcomplete basis set: A strategy employed by V1? *Vision Research*, 37, 1997.

[19] M. Ranzato, F.J. Huang, Y. Boureau, and Y. LeCun. Unsupervised Learning of Invariant Feature Hierarchies with Applications to Object Recognition. In *CVPR*, 2007.

[20] E. Shechtman and M. Irani. Matching Local Self-Similarities across Image and Videos. In *CVPR*, 2007.

[21] R. Tibshirani and J. Taylor. The Solution Path of the Generalized Lasso. *The Annals of Statistics*, 2010.

[22] J. Wang, J. Yang, K. Yu, F. Lv, T. Huang, and Y. Gong. Locality-constrained Linear Coding for Image Classification. In *CVPR*, 2010.

[23] J. Yang, J. Wright, T. Huang, and Y. Ma. Image Super-Resolution via Sparse Representation. *TIP*, 2010.

[24] J. Yang, K. Yu, Y. Gong, and T.S. Huang. Linear spatial pyramid matching using sparse coding for image classification. In *CVPR*, 2009.

[25] J. Yang, K. Yu, and T. Huang. Supervised Translation-Invariant Sparse Coding. In *CVPR*, 2010.

[26] K. Yu, T. Zhang, and Y. Gong. Nonlinear Learning using Local Coordinate Coding. In *NIPS*, 2009.

